# Non-iterative Estimation with Perturbed Gaussian Markov Processes

**Yunsong Huang**       **B. Keith Jenkins**
Signal and Image Processing Institute
Department of Electrical Engineering-Systems
University of Southern California
Los Angeles, CA 90089-2564
`{yunsongh,jenkins}@sipi.usc.edu`

## Abstract

We develop an approach for estimation with Gaussian Markov processes that imposes a smoothness prior while allowing for discontinuities. Instead of propagating information laterally between neighboring nodes in a graph, we study the posterior distribution of the hidden nodes as a whole—how it is perturbed by invoking discontinuities, or weakening the edges, in the graph. We show that the resulting computation amounts to feed-forward fan-in operations reminiscent of V1 neurons. Moreover, using suitable matrix preconditioners, the incurred matrix inverse and determinant can be approximated, without iteration, in the same computational style. Simulation results illustrate the merits of this approach.

## 1   Introduction

Two issues, (i) efficient representation, and (ii) efficient inference, are of central importance in the area of statistical modeling of vision problems. For generative models, often the ease of generation and the ease of inference are two conflicting features. Factor Analysis [1] and its variants, for example, model the input as a linear superposition of basis functions. While the generation, or synthesis, of the input is immediate, the inference part is usually not. One may apply a set of filters, *e.g.*, Gabor filters, to the input image. In so doing, however, the statistical modeling is only deferred, and further steps, either implicit or explicit, are needed to capture the 'code' carried by those filter responses. By characterizing mutual dependencies among adjacent nodes, Markov Random Field (MRF) [2] and graphical models [3] are other powerful ways for modeling the input, which, when continuous, is often conveniently assumed to be Gaussian. In vision applications, it's suitable to employ smoothness priors admitting discontinuities [4]. Examples include *weak membranes* and *plates* [5], formulated in the context of variational energy minimization. Typically, the inference for MRF or graphical models would incur lateral propagation of information between neighboring units [6]. This is appealing in the sense that it consists of only simple, local operations carried out in parallel. However, the resulting latency could undermine the plausibility that such algorithms are employed in human early vision inference tasks [7].

In this paper we take the weak membrane and plate as instances of Gaussian processes (GP). We show that the effect of marking each discontinuity (hereafter termed as "bond-

breaking") is to perturb the inverse of covariance matrix of the hidden nodes $x$ by a matrix of rank 1. When multiple bonds are broken, the computation of the posterior mean and covariance of $x$ would involve the inversion of a matrix, which typically has large condition number, implying very slow convergence in straight-forward iterative approaches. We show that there exists a family of preconditioners that can bring the condition number close to 1, thereby greatly speeding up the iteration—to the extent that a single step would suffice in practice. Therefore, the predominant computation employed in our approach is noniterative, of fan-in and fan-out style. We also devise ways to learn the parameters regarding state and observation noise non-iteratively. Finally, we report experimental results of applying the proposed algorithm to image-denoising.

## 2 Perturbing a Gaussian Markov Process (GMP)

Consider a spatially invariant GMP defined on a torus, $x \sim \mathcal{N}(0, Q_0)$, whose energy—defined as $x^T Q_0^{-1} x$—is the sum of energies of all edges[1] in the graph, due to the Markovian property. In what follows, we perturb the potential matrix $Q_0^{-1}$ by reducing the coupling energy of certain bonds[2]. This relieves the smoothness constraint on the nodes connected via those bonds.

Suppose the energy reduction of a bond connecting node $i$ and $j$ (whose state vectors are $x_i$ and $x_j$, respectively) can be expressed as $(x_i^T f_i + x_j^T f_j)^2$, where $f_i$ and $f_j$ are coefficient vectors. This becomes $(x^T f)^2$, if $f$ is constructed to be a vector of same size as $x$, with the only non-zero entries $f_i$ and $f_j$ corresponding to node $i$ and $j$. This manipulation can be identified with a rank-1 perturbation of $Q_0^{-1}$, as $Q_1^{-1} \leftarrow Q_0^{-1} - ff^T$, which is equivalent to $x^T Q_1^{-1} x \leftarrow x^T Q_0^{-1} x - (x^T f)^2, \forall x$. We call this an elementary perturbation of $Q_0^{-1}$, and $f$ an elementary perturbation vector associated with the particular bond.

When $L$ such perturbations have taken place (cf. Fig. 1), we form the $L$ perturbation vectors into a matrix $F_1 = [f^1, \ldots, f^L]$, and then the collective perturbations yield

$$Q_1^{-1} \;=\; Q_0^{-1} - F_1 F_1^T \tag{1}$$

$$\text{and thus} \quad Q_1 \;=\; Q_0 + Q_0 F_1 (I - F_1^T Q_0 F_1)^{-1} F_1^T Q_0, \tag{2}$$

which follows from the Sherman-Morrison-Woodbury Formula (SMWF).

### 2.1 Perturbing a membrane and a plate

In a membrane model [5], $x_i$ is scalar and the energy of the bond connecting $x_i$ and $x_j$ is $(x_i - x_j)^2/q$, where $q$ is a parameter denoting the variance of state noise. Upon perturbation, this energy is reduced to $\eta^2 (x_i - x_j)^2/q$, where $0 < \eta \ll 1$ ensures positivity of the energy. Then, the energy reduction is $(1 - \eta^2)(x_i - x_j)^2/q$, from which we can identify $f_i = \sqrt{(1 - \eta^2)/q}$ and $f_j = -f_i$.

In the case of a plate [5], $x_i = [u_i, u_{hi}, u_{vi}]^T$, in which $u_i$ represents the intensity, while $u_{hi}$ and $u_{vi}$ represent its gradient in the horizontal and vertical direction, respectively. We define the energy of a horizontal bond connecting node $j$ and $i$ as $E_0^{(-,i)} = (u_{vi} - u_{vj})^2/q + d^{(-,i)T} O^{-1} d^{(-,i)}$, where

$$d^{(-,i)} = \begin{bmatrix} u_i \\ u_{hi} \end{bmatrix} - \begin{bmatrix} 1 & 1 \\ 0 & 1 \end{bmatrix} \begin{bmatrix} u_j \\ u_{hj} \end{bmatrix} \quad \text{and} \quad O = q \begin{bmatrix} 1/3 & 1/2 \\ 1/2 & 1 \end{bmatrix},$$

the superscript $(-,i)$ representing horizontal bond to the left of node $i$. The first and second term of $E^{(-,i)}$ would correspond to $(\partial^2 u(h,v)/\partial h \partial v)^2/q$ and $(\partial^2 u(h,v)/\partial h^2)^2/q$, respectively, if $u(h,v)$ is a continuous function of $h$ and $v$ (cf. [5]). If $E_0^{(-,i)}$ is reduced to $E_1^{(-,i)} = [(u_{vi} - u_{vj})^2 + (u_{hi} - u_{hj})^2]/q$, $i.e.$, coupling between node $i$ and $j$ exists only through their gradient values, one can show that the energy reduction $E_0^{(-,i)} - E_1^{(-,i)} = [u_i - u_j - (u_{hi} + u_{hj})/2]^2 \cdot 12/q$. Taking the actual energy reduction to be $(1 - \eta^2)(E_0^{(-,i)} - E_1^{(-,i)})$, we can identify $f_i^{(-,i)} = \sqrt{12(1-\eta^2)/q}[1, -1/2, 0]^T$ and $f_j^{(-,i)} = \sqrt{12(1-\eta^2)/q}[-1, -1/2, 0]^T$, where $0 < \eta \ll 1$ ensures the positive definiteness of the resulting potential matrix. A similar procedure can be applied to a vertical bond in the plate, producing a perturbation vector $f^{(|,i)}$, whose components are zero everywhere except for $f_i^{(|,i)} = \sqrt{12(1-\eta^2)/q}[1, 0, -1/2]^T$ and $f_j^{(|,i)} = \sqrt{12(1-\eta^2)/q}[-1, 0, -1/2]^T$, for which node $j$ is the lower neighbor of node $i$.

One can verify that $x^T f = 0$ when the plate assumes the shape of a linear slope, meaning that this perturbation produces no energy difference in such a case. $(x^T f)^2$ becomes significant when the perturbed, or broken, bond associated with $f$ straddles across a step discontinuity of the image. Such an $f$ is thus related to edge detection.

## 2.2 Hidden state estimation

Standard formulae exist for the posterior covariance $K$ and mean $\hat{x}$ of $x$, given a noisy observation[3] $y = Cx + n$, where $n \sim \mathcal{N}(0, rI)$.

$$\hat{x}^\alpha = K_\alpha C^T y/r, \quad \text{and} \quad K_\alpha = [Q_\alpha^{-1} + C^T C/r]^{-1}, \tag{3}$$

for either the unperturbed ($\alpha = 0$) or perturbed ($\alpha = 1$) process. Thus,

$$
\begin{aligned}
K_1 &= [Q_0^{-1} + C^T C/r - F_1 F_1^T]^{-1}, \quad \text{following Eq. 3 and 1} \\
&= [K_0^{-1} - F_1 F_1^T]^{-1}, \\
&= K_0 + W_1 H_1^{-1} W_1^T, \quad \text{applying SMWF}, \tag{4}
\end{aligned}
$$

$$\text{where} \quad H_1 \triangleq I - F_1^T K_0 F_1, \quad \text{and} \quad W_1 \triangleq K_0 F_1 \tag{5}$$

$$
\begin{aligned}
\therefore \hat{x}^1 &= K_1 C^T y/r \\
&= K_0 C^T y/r + W_1 H_1^{-1} W_1^T C^T y/r = \hat{x}^0 + \hat{x}^c, \tag{6}
\end{aligned}
$$

$$\text{where} \quad \hat{x}^c \triangleq W_1 H_1^{-1} W_1^T C^T y/r,$$

$$= W_1 H_1^{-1} z^1, \quad \text{where} \quad z^1 = W_1^T C^T y/r \tag{7}$$

On a digital computer, the above computation can be efficiently implemented in the Fourier domain, despite the huge size of $K_\alpha$ and $Q_\alpha$. For example, $K_1$ equals $K_0$—a circulant matrix—plus a rank-$L$ perturbation (cf. Eq. 4). Since each column of $W_1$ is a spatially shifted copy of a prototypical vector, arising from breaking either a horizontal or a vertical bond, convolution can be utilized in computing $W_1^T C^T y$. The computation of $H_1^{-1}$ is deferred to Section 3. On a neural substrate, however, the computation can be implemented by inner-products in parallel. For instance, $z^1 r$ is the result of inner-products between the input $y$ and the feed-forward fan-in weights $CW$, coded by the dendrites of identical neurons, each situated at a broken bond. Let $v^1 = H_1^{-1} z^1$ be the responses of another layer of neurons. Then $C\hat{x}^c = CWv^1$ amounts to the back-projection of layer $v^1$ to the input plane with fan-out weights identical to the fan-in counterpart.

We can also apply the above procedure incrementally[4], $i.e.$, apply $F_1$ and then $F_2$, both consisting of a set of perturbation vectors. Quantities resulting from the $\alpha$'th perturba-

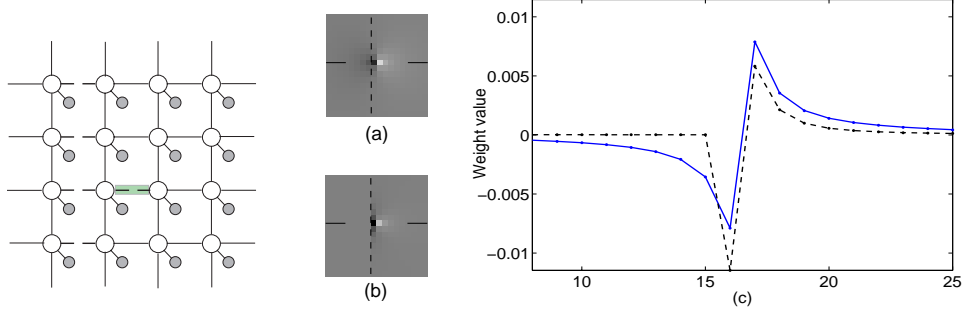

Figure 1: A portion of MRF. Solid and broken lines denote intact and broken bonds, respectively. Open circles denote hidden nodes $x_i$ and filled circles denote observed nodes $y_i$.

Figure 2: The resulting receptive field of the edge detector produced by breaking the shaded bond shown in Fig. 1. The central vertical dashed line in (a) and (b) marks the location of the vertical streak of bonds shown as broken in Fig. 1. In (a), those bonds are not actually broken; in (b), they are. In (c), a central horizontal slice of (a) is plotted as a solid curve and the counterpart of (b) as a dashed curve.

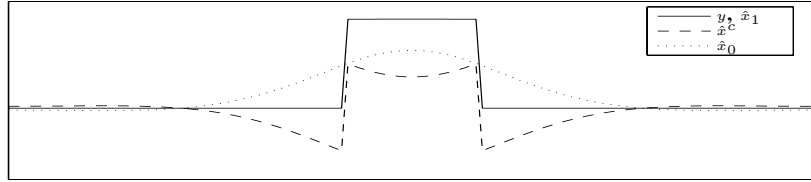

Figure 3: Estimation of $x$ given input $y$. $\hat{x}_0$: by unperturbed rod; $\hat{x}_1$: coinciding perfectly with $y$, is obtained by a rod whose two bonds at the step edges of $y$ are broken; $\hat{x}^c$: correction term, engendered by the perturbed rod.

tion step can be obtained from those of the $(\alpha - 1)$'th step, simply by replacing the subscript/superscript '1' and '0' with $\alpha$ and $\alpha - 1$, respectively, in Eqs. 1 to 6. In particular,

$$W_2 = K_1 F_2 = \underbrace{K_0 F_2}_{\widetilde{W_2}} + \underbrace{W_1 H_1^{-1} W_1^T F_2}_{\delta W_2}, \tag{8}$$

where $\widetilde{W_2}$ refers to the weights due to $F_2$ in the absence of perturbation $F_1$, which, when indeed existent, would exert a contextual effect on $F_2$, thereby contributing to the term $\delta W_2$.

Figure 2 illustrates this effect on one perturbation vector (termed 'edge detector') in a membrane model, wherein 'receptive field' refers to $\widetilde{W_2}$ and $W_2$ in the case of panel (a) and (b), respectively. Evidently, the receptive field of $W_2$ across the contextual boundary is pinched off. Figure 3 shows the estimation of $x$, cf. Eq. 6 and 7, using a 1D plate, *i.e.*, rod. We stress that once the relevant edges are detected, $\hat{x}^c$ is computed almost instantly, without the need of iterative refinement via lateral propagation. This could be related to the brightness filling-in signal[8].

## 2.3 Parameter estimation

As edge inference/detection is outside the scope of this paper, we limit our attention to finding optimal values for the parameters $r$ and $q$. Although the EM algorithm is possible

for that purpose, we strive for a non-iterative alternative. To that end, we reparameterize $r$ and $q$ into $r$ and $\varrho = q/r$. Given a possibly perturbed model $M_\alpha$, in which $x \sim \mathcal{N}(0, Q_\alpha)$, we have $y \sim \mathcal{N}(0, S_\alpha)$, where $S_\alpha = rI + CQ_\alpha C^T$. Note that $\widetilde{S_\alpha} \triangleq S_\alpha / r$ does not depend on $r$ when $\varrho$ is fixed, as $Q_\alpha \propto q \propto r \Longrightarrow S_\alpha \propto r$. Next, we aim to maximize the log-probability of $y$, which is a vector of $N$ components (or pixels).

$$
\begin{aligned}
\tilde{J}_\alpha \triangleq \mathrm{Ln}p(y|M_\alpha) &= -(N\mathrm{Ln}(2\pi) + \mathrm{Ln}|S_\alpha| + y^T S_\alpha^{-1} y)/2 \\
&= -(N\mathrm{Ln}(2\pi) + N\mathrm{Ln}r + \mathrm{Ln}|\widetilde{S_\alpha}| + (y^T \widetilde{S_\alpha}^{-1} y)/r)/2
\end{aligned}
$$

$$
\text{Setting} \quad \partial \tilde{J}_\alpha / \partial r = 0 \quad \Rightarrow \quad \hat{r} = E_\alpha / N, \quad \text{where} \quad E_\alpha \triangleq y^T \widetilde{S_\alpha}^{-1} y \tag{9}
$$

$$
\text{Define} \quad J \triangleq N\mathrm{Ln}E_\alpha + \mathrm{Ln}|\widetilde{S_\alpha}| = \text{const.} - 2\tilde{J}_\alpha|_{\hat{r}} \tag{10}
$$

$J$ is a function of $\varrho$ only, and we locate the $\hat{\varrho}$ that minimizes $J$ as follows. Prompted by the fact that $\varrho$ governs the *spatial scale* of the process [5] and scale channels exist in primate visual system, we compute $J(\varrho)$ for a preselected set of $\varrho$, corresponding to spatial scales half-octave apart, and then fit the resulting $J$'s with a cubic polynomial, whose location of minimum suggests $\hat{\varrho}$. We use this approach in Section 4.

Computing $J$ in Eq. 10 needs two identities, which are included here without proof (the second can be proven by using SMWF and its associated determinant identity): $E_\alpha = y^T(y - C\hat{x}^\alpha)$ (cf. Appendix A of [5]), and $|S_0|/|S_\alpha| = |B_\alpha|/|H_\alpha|$, where

$$
H_\alpha = I - F_\alpha^T K_0 F_\alpha, \quad \text{and} \quad B_\alpha \triangleq I - F_\alpha^T Q_0 F_\alpha \tag{11}
$$

That is, $E_\alpha$ can be readily obtained once $\hat{x}^\alpha$ has been estimated, and $|\widetilde{S_\alpha}| = |\widetilde{S_0}||H_\alpha|/|B_\alpha|$, in which $|\widetilde{S_0}|$ can be calculated in the spectral domain, as $S_0$ is circulant. The computation of $|H_\alpha|$ and $|B_\alpha|$ is dealt with in the next section.

## 3 Matrix Preconditioning

Some of the foregoing computation necessitates matrix determinant and matrix inverse, *e.g.*, $H^{-1}z^1$(cf. Eq. 7). Because $H$ is typically poorly conditioned, plain iterative means to evaluate $H^{-1}z^a$ would converge very slowly. Methods exist in the literature for finding a matrix $P$ ([9] and references therein) satisfying the following two criteria: (1) inverting $P$ is easy; (2) the condition number $\kappa(P^{-1}H)$ approaches 1. Ideally, $\kappa(P^{-1}H) = 1$ implies $P = H$. Here we summarize our findings regarding the best class of preconditioners when $H$ arises from some prototypical configurations of bond breaking. We call the following procedure Approximate Diagonalization (AD).

(1) 'DFT'. When a streak of broken bonds forms a closed contour, with a consistent polarity convention (*e.g.*, the excitatory region of the receptive field of the edge detector associated with each bond lies inside the enclosed region), $H$ and $B$ (cf. Eq. 11) are approximately circulant. Let $X$ be the unitary Fourier matrix of same size as $H$, then $H^e = X^\dagger H X$ would be approximately diagonal. Let $\Lambda_H$ be diagonal: $\Lambda_{Hij} = \delta_{ij}H^e{}_{ii}$, then $\widetilde{H} = X\Lambda_H X^\dagger$ is a circulant matrix approximating $H$; $\prod_i \Lambda_{Hii}$ approximates $|H|$; $X\Lambda_H^{-1}X^\dagger$ approximates $H^{-1}$. In this way, a computation such as $H^{-1}z^1$ becomes $X\Lambda_H^{-1}X^\dagger z^1$, which amounts to simple fan-in and fan-out operations, if we regard each column of $X$ as a fan-in weight vector. The quality of this preconditioner $\widetilde{H}$ can be evaluated by both the condition number $\kappa(\widetilde{H}^{-1}H)$ and the relative error between the inverse matrices:

$$
\epsilon \triangleq \|\widetilde{H}^{-1} - H^{-1}\|_F / \|H^{-1}\|_F, \tag{12}
$$

where $\|\,.\,\|_F$ denotes Frobenius norm. The same $X$ can approximately diagonalize $B$, and the product of the diagonal elements of the resulting matrix approximates $|B|$.

(2) 'DCST'. One end of the streak of broken bonds (target contour) abuts another contour, and the other end is open (*i.e.*, line-end). Imagine a vibrational mode of the membrane/plate given the configuration of broken bonds. The vibrational contrast of the nodes across the broken bond at a line-end has to be small, since in the immediate vicinity there exist paths of intact bonds linking the two nodes. This suggests a Dirichlet boundary condition at the line-end. At the abutting end (*i.e.*, a T-junction), however, the vibrational contrast can be large, since the nodes on different sides of the contour are practically decoupled. This suggests a von Neumann boundary condition. This analysis leads to using a transform (termed 'HSWA' in [10]) which we call 'DCST', denoting sine phase at the open end and cosine phase at the abutting end. The unitary transform matrix $X$ is given by: $X_{i,j} = 2\sqrt{2L+1}\cos(\pi(i-1/2)(j-1/2)/(L+1/2))$, $1 \le i,j \le L$, where $L$ is the number of broken bonds in the target contour.

(3) 'DST'. When the streak of broken bonds form an open-ended contour, $H$ can be approximately diagonalized by Sine Transform (cf. the intuitive rationale stated in case (2)), of which the unitary transform matrix $X$ is given by: $X_{i,j} = \sqrt{2/(L+1)}\sin(\pi ij/(L+1))$, $1 \le i,j \le L$.

For a 'clean' prototypical contour, the performance of such preconditioners is remarkable, typically producing $1 \le \kappa < 1.2$ and $\epsilon < 0.05$. When contours in the image are interconnected in a complex way, we first parse the image domain into non-overlapping enclosed regions, and then treat each region independently. A contour segment dividing two regions is shared between them, and thus would contribute two copies, each belonging to one region[11].

## 4   Experiment

We test our approach on a real image (Fig. 4a), which is corrupted with three increasing levels of white Gaussian noise: SNR = 4.79db (Fig. 4b), 3.52db, and 2.34db. Our task is to estimate the original image, along with finding optimal $q$ and $r$. We used both membrane and plate models, and in each case we used both the 'direct' method, which directly computes $H^{-1}$ in Eq. 7 and $|H|/|B|$ required in Eq. 10, and the 'AD' method, as described in Section 3, to compute those quantities in approximation.

We first apply a Canny detector to generate an edge map (Fig. 4g) for each noisy image, which is then converted to broken bonds. The large number (over $10^4$) of broken bonds makes the direct method impractical. In order to attain a 'direct' result, we partition the image domain into a $5 \times 5$ array of blocks (one such block is delineated by the inner square in Fig. 4g), and focus on each of them in turn by retaining edges not more than 10 pixels from the target block (this block's outer scope is delineated with the outer square in Fig. 4g). When $\hat{x}$ is inferred given this partial edge map, only its pixels within the block are considered valid and are retained. We mosaic up $\hat{x}$ from all those blocks to get the complete inferred image. In 'AD', we parse the contours in each block and apply different diagonalizers accordingly, as summarized in Section 3. The performance of the three types of AD is plotted in Fig. 5, from which it is evident that in majority of cases $\kappa < 1.5$ and $\epsilon \le 10\%$. Fig. 4e and f illustrate the procedure to find optimal $q/r$ for a membrane and a plate, respectively, as explained in Section 2.3. Note how good the cubic polynomial fit is, and that the results of AD do not deviate much from those of the direct (rigorous) method. Fig. 4c and 4d show $\hat{x}$ by a perturbed and intact membrane model, respectively. Notice that the edges, for instance around Lena's shoulder and her hat, in Fig. 4d are more smeared than those in Fig. 4c (cf. Fig. 3). Table 1 summarizes the value of optimal $q/r$ and Mean-Squared-Error (MSE). Our results compare favorably with those listed in the last column of the table, which is excerpted from [12].

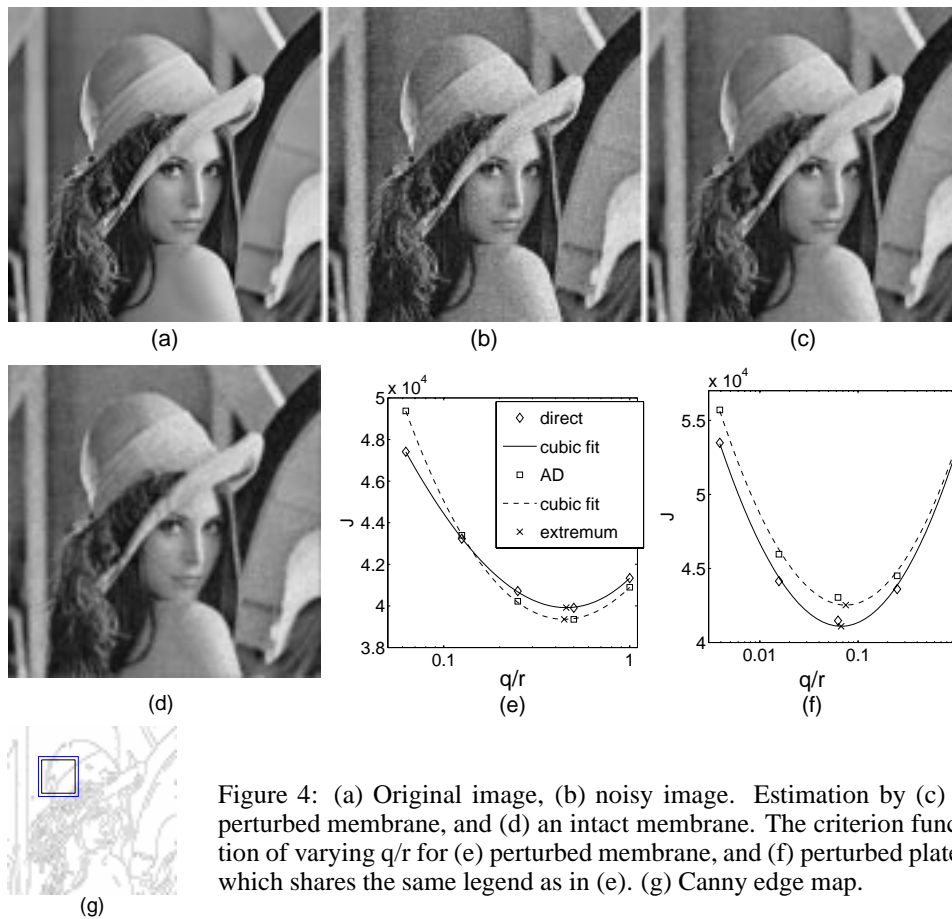

Figure 4: (a) Original image, (b) noisy image. Estimation by (c) a perturbed membrane, and (d) an intact membrane. The criterion function of varying q/r for (e) perturbed membrane, and (f) perturbed plate, which shares the same legend as in (e). (g) Canny edge map.

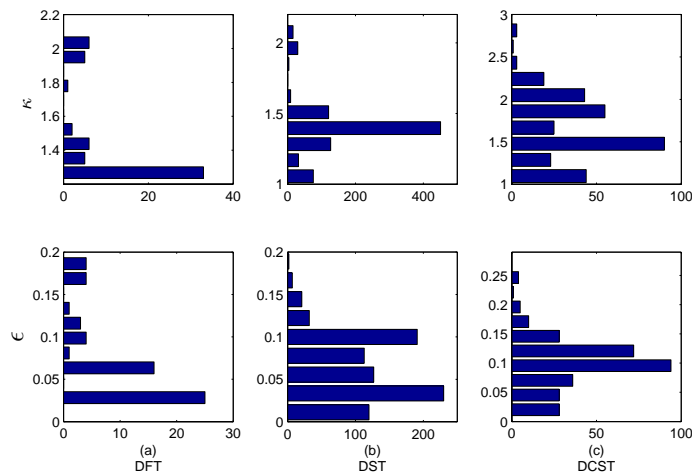

Figure 5: Histograms of condition number $\kappa$ after preconditioning, and relative error $\epsilon$ as defined in Eq. 12, illustrating the performance of preconditioners, DFT, DST, and DCST, on their respective datasets. Horizontal axes indicate the number of occurrences in each bin.

Table 1: Optimal q/r and MSE.

| SNR | membrane model | | | | plate model | | | | Improved Entropic [12] |
|---|---|---|---|---|---|---|---|---|---|
| | direct | | AD | | direct | | AD | | |
| | q/r | MSE | q/r | MSE | q/r | MSE | q/r | MSE | MSE |
| 4.79 | 0.456 | 92 | 0.444 | 92 | 0.067 | 100 | 0.075 | 98 | 121 |
| 3.52 | 0.299 | 104 | 0.311 | 104 | 0.044 | 111 | 0.049 | 108 | 138 |
| 2.34 | 0.217 | 115 | 0.233 | 115 | 0.033 | 119 | 0.031 | 121 | 166 |

## 5 Conclusions

We have shown how the estimation with perturbed Gaussian Markov processes—hidden state and parameter estimation—can be carried out in non-iterative way. We have adopted a holistic viewpoint. Instead of focusing on each individual hidden node, we have taken each process as an entity under scrutiny. This paradigm shift changes the way information is stored and represented—from the scenario where the global pattern of the process is embodied entirely by local couplings to the scenario where fan-in and fan-out weigths, in addition to local couplings, reflect the patterns of larger scales.

Although edge detection has not been treated in this paper, our formulation is capable of doing so, and our preliminary results are encouraging. It may be premature at this stage to translate the operations of our model to neural substrate; we speculate nevertheless that our approach may have relevance to understanding biological visual systems.

### Acknowledgments

This work was supported in part by the TRW Foundation, ARO (Grant Nos. DAAG55-98-1-0293 and DAAD19-99-1-0057), and DARPA (Grant No. DAAD19-0010356).

## Footnotes

[1]Henceforth called *bonds*, as *edge* will refer to intensity discontinuity in an image.

[2]The bond energy remains positive. This ensures the positive definiteness of the potential matrix.

[3] The observation matrix $C = I$ for a membrane, and $C = I \otimes [1, 0, 0]$ for a plate.

[4] Latency considerations, however, preclude the practicability of *fully* incremental computation.

## References

[1] Z. Ghahramani and M.J. Beal. Variational inference for Bayesian mixtures of factor analysers. In *Advances in Neural Information Processing Systems*, volume 12. MIT Press, 2000.

[2] S.Z. Li. *Markov Random Field Modeling in Computer Vision*. Springer-Verlag, 1995.

[3] M.I. Jordan, Z. Ghahramani, T.S. Jaakkola, and L.K. Saul. An introduction to variational methods for graphical models. *Machine Learning*, 37:183–233, 1999.

[4] F. C. Jeng and J. W. Woods. Compound Gauss-Markov random fields for image estimation. *IEEE Trans. on Signal Processing*, 39(3):683–697, 1991.

[5] A. Blake and A. Zisserman. *Visual Reconstruction*. MIT Press, 1987.

[6] J.S. Yedidia, W.T. Freeman, and Y. Weiss. Bethe free energy, kikuchi approximations, and belief propagation algorithms. Technical Report TR2001-16, MERL, May 2001.

[7] S. Thorpe, D. Fize, and C. Marlot. Speed of processing in the human visual system. *Nature*, 381:520–522, 1996.

[8] L. Pessoa and P. De Weerd, editors. *Filling-in: From Perceptual Completion to Cortical Reorganization*. Oxford: Oxford University Press, 2003.

[9] R. Chan, M. Ng, and C. Wong. Sine transform based preconditioners for symmetric toeplitz systems. *Linear Algebra and its Applications*, 232:237–259, 1996.

[10] S. A. Martucci. Symmetric convolution and the discrete sine and cosine transforms. *IEEE Trans. on Signal Processing*, 42(5):1038–1051, May 1994.

[11] H. Zhou, H. Friedman, and R. von der Heydt. Coding of border ownership in monkey visual cortex. *J. Neuroscience*, 20(17):6594–6611, 2000.

[12] A. Ben Hamza, H. Krim, and G. B. Unal. Unifying probabilistic and variational estimation. *IEEE Signal Processing Magazine*, pages 37–47, September 2002.
